# The Interplay of Symbolic and Subsymbolic Processes in Anagram Problem Solving

**David B. Grimes and Michael C. Mozer**
Department of Computer Science and Institute of Cognitive Science
University of Colorado, Boulder, CO 80309-0430 USA
{grimes,mozer}@cs.colorado.edu

## Abstract

Although connectionist models have provided insights into the nature of perception and motor control, connectionist accounts of higher cognition seldom go beyond an implementation of traditional symbol-processing theories. We describe a connectionist constraint satisfaction model of how people solve anagram problems. The model exploits statistics of English orthography, but also addresses the interplay of subsymbolic and symbolic computation by a mechanism that extracts approximate symbolic representations (partial orderings of letters) from subsymbolic structures and injects the extracted representation back into the model to assist in the solution of the anagram. We show the computational benefit of this extraction-injection process and discuss its relationship to conscious mental processes and working memory. We also account for experimental data concerning the difficulty of anagram solution based on the orthographic structure of the anagram string and the target word.

Historically, the mind has been viewed from two opposing computational perspectives. The symbolic perspective views the mind as a symbolic information processing engine. According to this perspective, cognition operates on representations that encode logical relationships among discrete symbolic elements, such as stacks and structured trees, and cognition involves basic operations such as means-ends analysis and best-first search. In contrast, the subsymbolic perspective views the mind as performing statistical inference, and involves basic operations such as constraint-satisfaction search. The data structures on which these operations take place are numerical vectors.

In some domains of cognition, significant progress has been made through analysis from one computational perspective or the other. The thesis of our work is that many of these domains might be understood more completely by focusing on the *interplay* of subsymbolic and symbolic information processing. Consider the higher-cognitive domain of problem solving. At an abstract level of description, problem solving tasks can readily be formalized in terms of symbolic representations and operations. However, the neurobiological hardware that underlies human cognition appears to be subsymbolic—representations are noisy and graded, and the brain operates and adapts in a continuous fashion that is difficult to characterize in discrete symbolic terms. At some level—between the computational level of the task description and the implementation level of human neurobiology—the symbolic and subsymbolic accounts must come into contact with one another. We focus on this point of contact by proposing mechanisms by which symbolic representations can modulate subsymbolic processing, and mechanisms by which subsymbolic representations

are made symbolic. We conjecture that these mechanisms can not only provide an account for the interplay of symbolic and subsymbolic processes in cognition, but that they form a sensible computational strategy that outperforms purely subsymbolic computation, and hence, symbolic reasoning makes sense from an evolutionary perspective.

In this paper, we apply our approach to a high-level cognitive task, anagram problem solving. An anagram is a nonsense string of letters whose letters can be rearranged to form a word. For example, the solution to the anagram puzzle RYTEHO is THEORY. Anagram solving is a interesting task because it taps higher cognitive abilities and issues of awareness, it has a tractable state space, and interesting psychological data is available to model.

# 1  A Subsymbolic Computational Model

We start by presenting a purely subsymbolic model of anagram processing. By subsymbolic, we mean that the model utilizes only English orthographic statistics and does not have access to an English lexicon. We will argue that this model proves insufficient to explain human performance on anagram problem solving. However, it is a key component of a hybrid symbolic-subsymbolic model we propose, and is thus described in detail.

## 1.1  Problem Representation

A computational model of anagram processing must represent letter orderings. For example, the model must be capable of representing a solution such as <THEORY>, or any permutation of the letters such as <RYTEHO>. (The symbols "<" and ">" will be used to delimit the beginning and end of a string, respectively.) We adopted a representation of letter strings in which a string is encoded by the set of letter pairs (hereafter, *bigrams*) contained in the string; for example, the bigrams in <THEORY> are: <T, TH, HE, EO, OR, RY, and Y>. The delimiters < and > are treated as ordinary symbols of the alphabet. We capture letter pairings in a *symbolic letter-ordering matrix*, or *symbolic ordering* for short. Figure 1(a) shows the matrix, in which the rows indicate the first letter of the bigram, and the columns indicate the second. A cell of the matrix contains a value of 1 if the corresponding bigram is present in the string. (This matrix formalism and all procedures in the paper can be extended to handle strings with repeated letters, which we do not have space to discuss.) The matrix columns and rows can be thought of as consisting of all letters from A to Z, along with the delimiters < and >. However, in the Figure we have omitted rows and columns corresponding to letters not present in the anagram. Similarly, we have omitted the < from the column space and the > from row space, as they could not by definition be part of any bigram. The seven bigrams indicated by the seven ones in the Figure uniquely specify the string THEORY.

As we've described the matrix, cells contain the truth value of the proposition that a particular bigram appears in the string being represented. However, the cell values have an interesting alternative interpretation: as the *probability* that a particular bigram is present. Figure 1(b) illustrates a matrix of this sort, which we call a *subsymbolic letter ordering matrix*, or *subsymbolic ordering* for short. In the Figure, the bigram TH occurs with probability 0.8. Although the symbolic orderings are obviously a subset of the subsymbolic orderings, the two representations play critically disparate roles in our model, and thus are treated as separate entities.

To formally characterize symbolic and subsymbolic ordering matrices, we define a *mask* vector, $\mu$, having $N = 28$ elements, corresponding to the 26 letters of the alphabet plus the two delimiters. Element $i$ of the mask, $\mu_i$, is set to one if the corresponding letter appears in the anagram string and zero if it does not. In both the symbolic and subsymbolic orderings, the matrices are constrained such that elements in row $i$ and column $i$ must sum

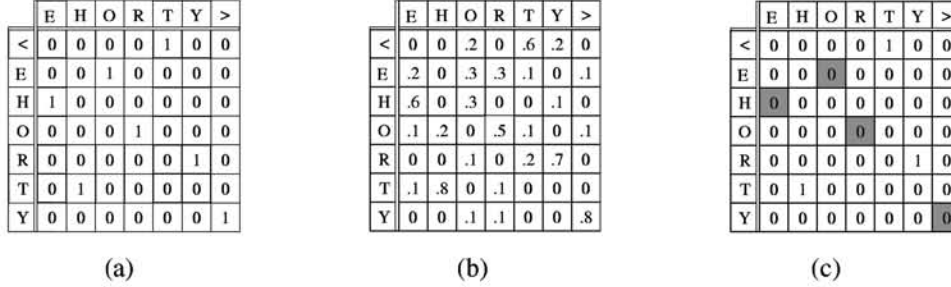

(a)           (b)           (c)

Figure 1: (a) A symbolic letter-ordering matrix for the string THEORY. (b) A subsymbolic letter-ordering matrix whose cells indicate the probabilities that particular bigrams are present in a letter string. (c) A symbolic partial letter-ordering matrix, formed from the symbolic ordering matrix by setting to zero a subset of the elements, which are highlighted in grey. The resulting matrix represents the partial ordering { <TH, RY }.

to $\mu_i$. If one extracts all rows and columns for which $\mu_i = 1$ from a symbolic ordering, as we have done in Figure 1(a), a permutation matrix is obtained. If one extracts all rows and columns for which $\mu_i = 1$ from a subsymbolic ordering, as we have done in Figure 1(b), the resulting matrix is known as *doubly stochastic*, because each row and column vector can each be interpreted as a probability distribution.

## 1.2 Constraint Satisfaction Network

A simple computational model can be conceptualized by considering each cell in the subsymbolic ordering matrix to correspond to a standard connectionist unit, and to consider each cell value as the activity level of the unit. In this conceptualization, the goal of the connectionist network is to obtain a pattern of activity corresponding to the solution word, given the anagram. We wish for the model to rely solely on orthographic statistics of English, avoiding lexical knowledge at this stage. Our premise is that an interactive model—a model that allows for top-down lexical knowledge to come in contact with the bottom-up information about the anagram—would be too powerful; i.e., the model would be super-human in its ability to identify lexical entries containing a target set of letters. Instead, we conjecture that a suitable model of human performance should be primarily bottom-up, attempting to order letters without the benefit of the lexicon. Of course, the task cannot be performed without a lexicon, but we defer discussion of the role of the lexicon until we first present the core connectionist component of the model.

The connectionist model is driven by three constraints: (1) solutions should contain bigrams with high frequency in English, (2) solutions should contain trigrams with high frequency in English, and (3) solutions should contain bigrams that are consistent with the bigrams in the original anagram. The first two constraints attempt to obtain English-like strings. The third constraint is motivated by the observation that anagram solution time depends on the arrangement of letters in the original anagram (e.g., Mayzner & Tresselt, 1959). The three constraints are embodied by a constraint-satisfaction network with the following *harmony function*:

$$H = \sum_{ij} \beta_{ij} p_{ij} + \omega \sum_{ijk} \tau_{ijk} p_{ij} p_{jk} + \xi \sum_{ij} p_{ij} s_{ij} \qquad (1)$$

where $p_{ij}$ denotes the value of the cell corresponding to bigram $ij$, $\beta_{ij}$ is monotonically related to the frequency of bigram $ij$ in English, $\tau_{ijk}$ is monotonically related to the frequency of trigram $ijk$ in English, $s_{ij}$ is 1 if the original anagram contained bigram $ij$ or 0 otherwise, and $\omega$ and $\xi$ are model parameters that specify the relative weighting of the trigram and unchanged-ordering constraints, respectively.

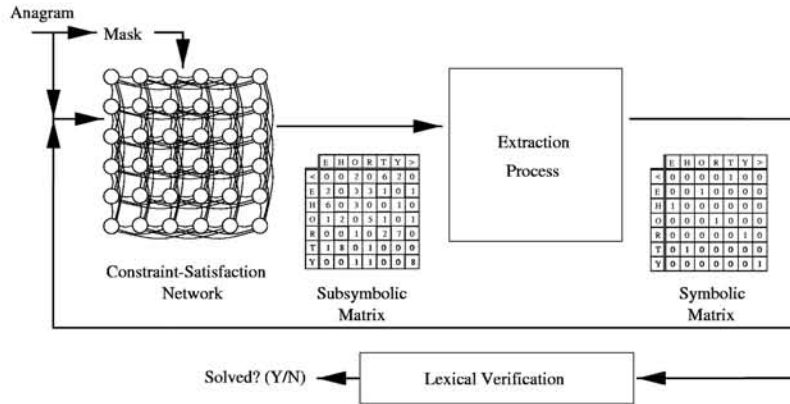

Figure 2: The Iterative Extraction-Injection Model

The harmony function specifies a measure of goodness of a given matrix in terms of the degree to which the three sets of constraints are satisfied. Running the connectionist network corresponds to searching for a local optimum in the harmony function. The local optimum can be found by gradient ascent, i.e., defining a unit-update rule that moves uphill in harmony. Such a rule can be obtained via the derivative of the harmony function: $\Delta p_{ij} = \epsilon \frac{\partial H}{\partial p_{ij}}$.

Although the update rule ensures that harmony will increase over time, the network state may violate the conditions of the doubly stochastic matrix by allowing the $p_{ij}$ to take on values outside of $[0, 1]$, or by failing to satisfy the row and column constraints. The procedure applied to enforce the row and column constraints involves renormalizing the activities after each harmony update to bring the activity pattern arbitrarily close to a doubly-stochastic matrix. The procedure, suggested by Sinkhorn (1964), involves alternating row and column normalizations (in our case to the values of the mask vector). Sinkhorn proved that this procedure will asymptotically converge on a doubly stochastic matrix. Note that the Sinkhorn normalization procedure must operate at a much finer time grain than the harmony updates, in order to ensure that the updates do not cause the state to wander from the space of doubly stochastic matrices.

## 2 The Iterative Extraction-Injection Model

The constraint-satisfaction network we just described is inadequate as a model of human anagram problem solving for two principle reasons. First, the network output generally does not correspond to a symbolic ordering, and hence has no immediate interpretation as a letter string. Second, the network has no access to a lexicon so it cannot possibly determine if a candidate solution is a word. These two concerns are handled by introducing additional processing components to the model. The components—called *extraction*, *verification*, and *injection*—bring subsymbolic representations of the constraint-satisfaction network into contact with the symbolic realm.

The extraction component converts a subsymbolic ordering—the output of the constraint-satisfaction network—into a symbolic ordering. This symbolic ordering serves as a candidate solution to the anagram. The verification component queries the lexicon to retrieve words that match or are very close to the candidate solution. If no lexical item is retrieved that can serve as a solution, the injection component feeds the candidate solution back

into the constraint-satisfaction network in the form of a bias on subsequent processing, in exactly the same way that the original anagram did on the first iteration of constraint satisfaction.

Figure 2 shows a high-level sketch of the complete model. The intuition behind this architecture is as follows. The symbolic ordering extracted on one iteration will serve to constrain the model's interpretation of the anagram on the next iteration. Consequently, the feedback forces the model down one path in a solution tree. When viewed from a high level, the model steps through a sequence of symbolic states. The transitions among symbolic states, however, are driven by the subsymbolic constraint-satisfaction network. To reflect the importance of the interplay between symbolic and subsymbolic processing, we call the architecture the *iterative extraction-injection model*.

Before describing the extraction, verification, and injection components in detail, we emphasize one point about the role of the lexicon. The model makes a strong claim about the sort of knowledge used to guide the solution of anagrams. Lexical knowledge is used only for verification, not for generation of candidate solutions. The limited use of the lexicon restricts the computational capabilities of the model, but in a way that we conjecture corresponds to human limitations.

## 2.1  Symbolic Extraction

The extraction component transforms the subsymbolic ordering matrix to an approximately equivalent symbolic ordering matrix. In essence, the extraction component treats the network activities as probabilities that pairs of letters will be joined, and samples a symbolic matrix from this probability distribution, subject to the restriction that each letter can precede or follow at most one other letter.

If subsymbolic matrix element $p_{ij}$ has a value close to 1, then it is clear that bigram $ij$ should be included in the symbolic ordering. However, if a row or column of a subsymbolic ordering matrix is close to uniform, the selection of a bigram in that row or column will be somewhat arbitrary. Consequently, we endow the model with the ability to select only some bigrams and leave other letter pairings unspecified. Thus, we allow the extraction component to consider symbolic *partial orderings*—i.e., a subset of the letter pairings in a complete ordering. For example, { <TH, RY } is a partial ordering that specifies that the T and H belong together in sequence at the beginning of the word, and the R should precede the Y, but does not specify the relation of these letter clusters to one another or to other letters of the anagram. Formally, a symbolic partial ordering matrix is a binary matrix in which the row and columns sum to values less than or equal to the corresponding mask value. A symbolic partial ordering can be formed by setting to zero some elements of a symbolic ordering (Figure 1(c)).

In the context of this task, a subsymbolic ordering is best viewed as a set of parameters specifying a distribution over a space $\mathcal{P}$ of all possible symbolic partial ordering matrices. Rather than explicitly generating and assigning probabilities to each element in $\mathcal{P}$, our approach samples from the distribution specified by the subsymbolic ordering using Markov Chain Monte Carlo (Neal, 1993). Our MCMC method obtains samples consistent with the bigram probabilities $p_{ij}$ and the row and column constraints, $\mu_j$.

## 2.2  Lexical Verification

Lexical verification involves consulting the lexicon to identify and validate candidate solutions. The extracted symbolic partial ordering is fed into the lexical verification component to identify a set of words, each of which is *consistent* with the partial ordering. By consistent, we mean the word contains all of the bigrams in the partial ordering. This set of words

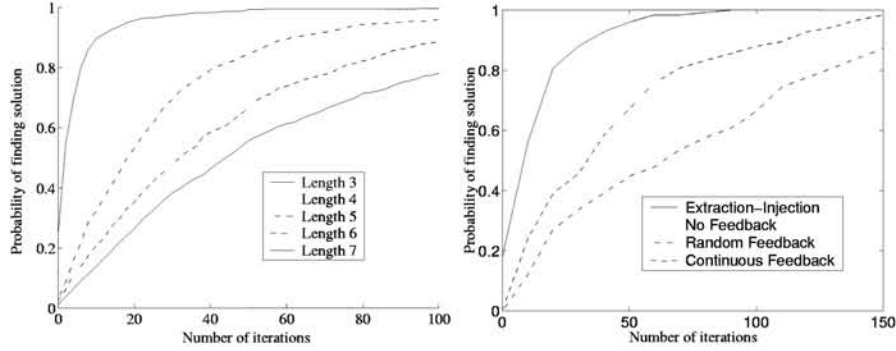

Figure 3: (a) Probability of finding solution for different word lengths as a function of number of iterations. (b) Convergence of the extraction-injection model and variants of the feedback mechanism.

is then checked to see if any word contains the same letters as the anagram. If so, the lexical verifier returns that the problem is solved. Otherwise, the lexical verifier indicates failure. Because the list of consistent words can be extremely large, and recalling and processing a large number of candidates seems implausible, we limit the size of the consistent set by introducing a recall parameter $\eta$ that controls the maximum size of the consistent set. If the actual number of consistent words is larger, a random sample of size $\eta$ is retrieved.

## 2.3   Injection

When the lexical verification component fails, the symbolic partial ordering is injected into the constraint-satisfaction network, replacing the letter ordering of the original anagram, and a new processing iteration begins. Were it not for new bias injected into the constraint satisfaction network, the constraint-satisfaction network would produce the same output as on the previous iteration, and the model would likely become stuck without finding a solution. In our experiments, we show that injecting the symbolic partial ordering allows the model to arrive at a solution more rapidly than other sorts of feedback.

## 3   Results and Discussion

Through simulation of our architecture we modeled several basic findings concerning human anagram problem solving. In our simulations, we define the *model solution time* to be the number of extraction-injection iterations before the solution word is identified.

Figure 3(a) shows the probability of the model finding the a solution as a function of the number of iterations the model is allowed to run and the number of letters in the word set. The data set consists of 40 examples for each of five different word lengths. The most striking result is that the probability of finding a solution increases monotonically over time. It is also interesting to note that the model's asymptotic accuracy is 100%, indicating that the model is computationally sufficient to perform the task. Of more significance is the fact that the model exhibits the word length effect as reported in Sargent (1940), that is, longer words take more time to solve.

Our model can explain other experimental results on anagram problem solving. Mayzner and Tresselt (1958) found that subjects were faster to find solutions composed of high frequency bigrams than solutions composed of low frequency bigrams. For example, SHIN contains higher frequency bigrams than HYMN. The iterative extraction-injection model reproduced this effect in the solution time to two classes of five five-letter words. Each

word was presented 30 times to obtain a distribution of solution times. A mean of 5.3 iterations was required for solutions composed of high frequency bigrams, compared to a mean of 21.2 iterations for solutions composed of low frequency bigrams. The difference is statistically reliable ($F(1,8) = 30.3, p < .001$). It is not surprising that the model produces this result, as the constraint-satisfaction network attempts to generate high frequency pairings of letters.

Mayzner and Tresselt (1959) found that subjects also are faster to solve an anagram if the anagram is composed of low frequency bigrams. For example, RCDA might be recognized as CARD more readily than would DACR. Our model reproduces this result as well. We tested the model with 25 four-letter target words whose letters could be rearranged to form anagrams with either low or high bigram frequency; each target word was presented 30 times. The mean solution time for low bigram-frequency anagrams was 21.4, versus 27.6 for high bigram-frequency anagrams. This difference is statistically reliable ($F(1,24) = 41.4, p < .001$). The difference is explained by the model's initial bias to search for solutions containing bigrams in the anagram, plus the fact that the model has a harder time pulling apart bigrams with high frequency.

Simulation results to date have focused on the computational properties of the model, with the goal of showing that the iterative extraction-injection process leads to efficient solution times. The experiments involve testing performance of models with some aspect of the iterative extraction-injection model modified. Three such variants were tested: 1) the feedback connection was removed, 2) random symbolic partial orderings were fed-back, and 3) subsymbolic partial orderings were fed-back. The experiment consisted of 125 words taken from Kucera and Francis (1967) corpus, which was also used for bigram and trigram frequencies. The median of 25 solution times for each word/model was used to compute the mean solution time for the original, no feedback, random feedback, and continuous feedback: 13.43, 41.88, 74.91, 43.17. The key result is that the iterative extraction-injection model was reliably 3-5 faster than the variants, as respective $F(1, 124, p < 0.001)$ scores were 87.8, 154.3, 99.1. Figure 3(b) shows the probability that each of these four models found the solution at a given time.

Although our investigation of this architecture is just beginning, we have shown that the model can explain some fundamental behavioral data, and that surprising computational power arises from the interplay of symbolic and subsymbolic information processing.

## Acknowledgments

This work benefited from the initial explorations and ideas of Tor Mohling. This research was supported by Grant 97-18 from the McDonnell-Pew Program in Cognitive Neuroscience, and by NSF award IBN-9873492.

## References

Kucera, H. & Francis, W. N. (1967). *Computational analysis of present-day American English.* Providence, RI: Brown University Press.

Mayzner, M. S. & Tresselt, M. E. (1958). *Anagram solution times: A function of letter and word frequency.* Journal of Experimental Psychology, 56, 376-379.

Mayzner, M. S. & Tresselt, M. E. (1959). *Anagram solution times: A function of transitional probabilities.* Journal of Experimental Psychology, 63, 510-513.

Neal, R. M. (1993). *Probabilistic inference using Markov Chain Monte Carlo Methods.* Technical Report CRG-TR-93-1, Dept. of Computer Science, University of Toronto.

Sargent, S. Stansfeld (1940). *Thinking Processes at Various Levels of Difficulty.* Archives of Psychology 249. New York.

Sinkhorn, Richard (1964). *A Relationship Between Arbitrary Positive Matrices and Doubly Stochastic Matrices.* Annals of Mathematical Statistics, Vol. 35, No. 2. pp. 876-879.
